# Timing and Partial Observability in the Dopamine System

**Nathaniel D. Daw**[1,3]**, Aaron C. Courville**[2,3]**, and David S. Touretzky**[1,3]

[1]Computer Science Department, [2]Robotics Institute, [3]Center for the Neural Basis of Cognition
Carnegie Mellon University, Pittsburgh, PA 15213
{daw,aaronc,dst}@cs.cmu.edu

## Abstract

According to a series of influential models, dopamine (DA) neurons signal reward prediction error using a temporal-difference (TD) algorithm. We address a problem not convincingly solved in these accounts: how to maintain a representation of cues that predict delayed consequences. Our new model uses a TD rule grounded in partially observable semi-Markov processes, a formalism that captures two largely neglected features of DA experiments: hidden state and temporal variability. Previous models predicted rewards using a tapped delay line representation of sensory inputs; we replace this with a more active process of inference about the underlying state of the world. The DA system can then learn to map these inferred states to reward predictions using TD. The new model can explain previously vexing data on the responses of DA neurons in the face of temporal variability. By combining statistical model-based learning with a physiologically grounded TD theory, it also brings into contact with physiology some insights about behavior that had previously been confined to more abstract psychological models.

## 1 Introduction

A series of models [1, 2, 3, 4, 5] based on temporal-difference (TD) learning [6] has explained most responses of primate dopamine (DA) neurons during conditioning [7] as an error signal for predicting reward, and has also identified the DA system as a substrate for conditioning behavior [8]. We address a troublesome issue from these models: how to maintain a representation of cues that predict delayed consequences. For this, we use a formalism that extends the Markov processes in which previous models were grounded.

Even in the laboratory, the world is often poorly described as Markov in immediate sensory observations. In trace conditioning, for instance, nothing observable spans the delay between a transient stimulus and the reward it predicts. For DA models, this raises problems of coping with hidden state and of tracking temporal intervals. Most previous models address these issues using a tapped delay line representation of the world's state. This augments the representation of current sensory observations with remembered past observations, dividing temporal intervals into a series of states to mark the passage of time. But linear combinations of tapped delay lines do not properly model variability in the intervals between events. Also, the augmented representation may poorly match the contingency

structure of the experimental situation: for instance, depending on the amount of history retained, it may be insufficient to span delays, or it may contain old, irrelevant data.

We propose a model that better reflects experimental situations by using a formalism that explicitly incorporates hidden state and temporal variability: a partially observable semi-Markov process. The proposal envisions the interaction between a cortical perceptual system that infers the world's hidden state using an internal world model, and a dopaminergic TD system that learns reward predictions for these inferred states. This model improves on its predecessors' descriptions of neuronal firing in situations involving temporal variability, and suggests additional connections with animal behavior.

## 2  DA models and temporal variability

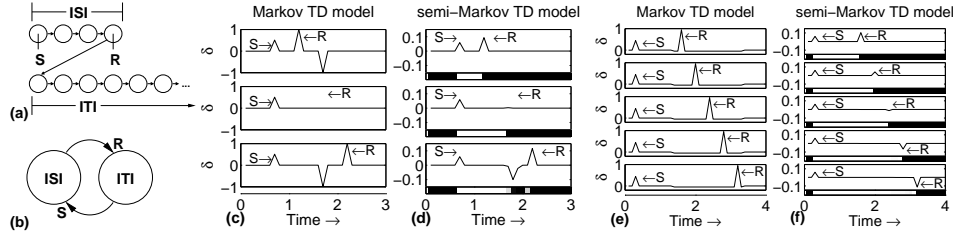

Figure 1: S: stimulus; R: reward. (a,b) State spaces for the Markov tapped delay line (a) and our semi-Markov (b) TD models of a trace conditioning experiment. (c,d) Modeled DA activity (TD error) when an expected reward is delivered early (top), on time (middle) or late (bottom). The tapped delay line model (c) produces spurious negative error after an early reward, while, in accord with experiments, our semi-Markov model (d) does not. Shaded stripes under (d) and (f) track the model's belief distribution over the world's hidden state (given a one-timestep backward pass), with the ISI in white, the ITI in black, and gray for uncertainty between the two. (e,f) Modeled DA activity when reward timing varies uniformly over a range. The tapped delay line model (e) incorrectly predicts identical excitation to rewards delivered at all times, while, in accord with experiment, our model (f) predicts a response that declines with delay.

Several models [1, 2, 3, 4, 5] identify the firing of DA neurons with the reward prediction error signal $\delta_t$ of a TD algorithm [6]. In the models, DA neurons are excited by positive error in reward prediction (caused by unexpected rewards or reward-predicting stimuli) and inhibited by negative prediction error (caused by the omission of expected reward). If a reward arrives as expected, the models predict no change in firing rate. These characteristics have been demonstrated in recordings of primate DA neurons [7]. In idealized form (neglecting some instrumental contingencies), these experiments and the others that we consider here are all variations on trace conditioning, in which a phasic stimulus such as a flash of light signals that reward will be delivered after a delay.

TD systems map a representation of the state of the world to a prediction of future reward, but previous DA modeling exploited few experimental constraints on the form of this representation. Houk et al. [1] computed values using only immediately observable stimuli and allowed learning about rewards to accrue to previously observed stimuli using eligibility traces. But in trace conditioning, DA neurons show a timed pause in their background firing when an expected reward fails to arrive [7]. Because the Houk et al. [1] model does not learn temporal relationships, it cannot produce well timed inhibition. Montague et al. [2] and Schultz et al. [3] addressed these data using a tapped delay line representation of stimulus history [8]: at time $t$, each stimulus is represented by a vector whose $n$th element

codes whether the stimulus was observed at time $t - n$. This representation allows the models to learn the temporal relationship between stimulus and reward, and to correctly predict phasic inhibition timelocked to omitted rewards.

These models, however, mispredict the behavior of DA neurons when the interval between stimulus and reward varies. In one experiment [9], animals were trained to expect a constant stimulus-reward interval, which was later varied. When a reward is delivered earlier than expected, the tapped delay line models correctly predict that it should trigger positive error (dopaminergic excitation), but also incorrectly predict a further burst of *negative* error (inhibition, not seen experimentally) when the reward fails to arrive at the time it was originally expected (Figure 1c, top). In part, this occurs because the models do not represent the reward as an observation, so its arrival can have no effect on later predictions. More fundamentally, this is a problem with how the models partition events into a state space.

Figure 1a illustrates how the tapped delay lines mark time in the interval between stimulus and reward using a series of states, each of which learns its own reward prediction. After the stimulus occurs, the model's representation marches through each state in succession. But this device fails to capture a *distribution* over the interval between two events. If the second event has occurred, the interval is complete and the system should not expect reward again, but the tapped delay line continues to advance. This may be correctable, though awkwardly, by representing the reward with its own delay line, which can then learn to suppress further reward expectation after a reward occurs [10]. However, to our knowledge it is experimentally unclear whether the suppression of this response requires repeated experience with the situation, as this account predicts. Also, whether this works depends on how information from multiple cues is combined into an aggregate reward prediction (i.e. on the function approximator used: it is easy to verify that a standard linear combination of the delay lines does not suffice).

The models have a similar problem with a related experiment [11] (Figure 1e) where the stimulus-reward interval varied uniformly over a range of delays throughout training. In this case, all substates within the interval see reward with the same (low) probability, so each produces identical positive error when reward occurs there. In animal experiments, however, stronger dopaminergic activity is seen for earlier rewards [11].

## 3 A new model

Both of these experiments demonstrate that current TD models of DA do not adequately treat variability in event timing. We address them with a TD model grounded in a formalism that incorporates temporal variability, a partially observable [12] semi-Markov [13] process. Such a process is described by three functions, $O$, $Q$, and $D$, operating over two sets: the hidden states $\mathcal{S}$ and observations $\mathcal{O}$. $Q$ associates each state with a probability distribution over possible successors. If the process is in state $s \in \mathcal{S}$, then the next state is $s'$ with probability $Q_{ss'}$. These discrete state transitions can occur irregularly in continuous time (which we approximate to arbitrarily fine discretization). The dwell time $\tau$ spent in $s$ before making a transition is distributed with probability $D_{s\tau}$; we define the indicator $\phi_t$ as one if the state transitioned between $t$ and $t + 1$ and zero otherwise. On entering $s$, the process emits some observation $o \in \mathcal{O}$ with probability $O_{so}$. Some observations are distinguished as rewarding; we separately write the reward magnitude of an observation as $r$. Note that the processes we consider in this paper do not contain decisions.

In this formalism, a trace conditioning experiment can be treated as alternation between two states (Figure 1b). The states correspond to the intervals between stimulus and reward (interstimulus interval: ISI) and between reward and stimulus (intertrial interval: ITI). A stimulus is the likely observation when entering the ISI and a reward when entering the ITI.

We will index variables both by the time $t$ and by a discrete index $n$ which counts state transitions; e.g. the $n$th state, $\mathbf{s}_n$, is entered at time $t = \sum_{k=1}^{n-1} \tau_k$ and can thus also be written as $\mathbf{s}_t$. If $\phi_t = 0$ (if the state did not transition between $t$ and $t+1$) then $\mathbf{s}_{t+1} = \mathbf{s}_t$, $\mathbf{o}_{t+1}$ is null and $\mathbf{r}_{t+1} = 0$ (i.e., nonempty observations and rewards occur only on transitions). State transitions may be *unsignaled*: $\mathbf{o}_{t+1}$ may be null even if $\phi_t = 1$. An unsignaled transition into the ITI state occurs in our model when reward is omitted, a common experimental manipulation [7]. This example demonstrates the relationship between temporal variability and partial observability: if reward timing can vary, nothing in the observable state reveals whether a late reward is still coming or has been omitted completely.

TD algorithms [6] approximate a function mapping each state to its *value*, defined as the expectation (with respect to variability in reward magnitude, state succession, and dwell times) of summed, discounted future reward, starting from that state. In the semi-Markov case [13], a state's value is defined as the reward expectation at the moment it is entered; we do not count rewards received on the transition in. The value of the $n$th state entered is:

$$
\begin{aligned}
V_{s_n} &= E\left[\gamma^{\tau_n} \mathbf{r}_{n+1} + \gamma^{\tau_n + \tau_{n+1}} \mathbf{r}_{n+2} + ...\right] \\
&= E\left[\gamma^{\tau_n}(\mathbf{r}_{n+1} + V_{s_{n+1}})\right]
\end{aligned}
$$

where $\gamma < 1$ is a discounting parameter.

We address partial observability by using model-based inference to determine a distribution over the hidden states, which then serves as a basis over which a modified TD algorithm can learn values. The approach is similar to the Q-learning algorithm of Chrisman [14]. In our setting, however, values can in principle be learned exactly, since without decisions, they are linear in the space of hidden states.

For state inference, we assume that the brain's sensory processing systems use an internal *model* of the semi-Markov process — that is, the functions $O$, $Q$, and $D$. Here we take the model as given, though we have treated parts of the problem of learning such models elsewhere [15]. A key assumption about this internal model is that its distributions over intervals, rewards and observations contain *asymptotic uncertainty*, that is, they are not arbitrarily sharp. When learning internal models, such uncertainty can result from an assumption that parameters of the world are constantly changing [16]. Thus, in the inference model for the trace conditioning experiment, the ISI duration is modeled with a probability distribution with some nonzero variance rather than an impulse function. The model likewise assigns a small probability to anomalous transitions and observations (e.g. unrewarded transitions into the ITI state). This uncertainty is present only in the internal model: most anomalous events never occur in our simulations.

Given the model and a series of observations $\mathbf{o}_1 \ldots \mathbf{o}_t$, we can determine the likelihood that each hidden state is active using a standard forward-backward algorithm for hidden semi-Markov models [17]. The important quantity is the probability, for each state, that the system left that state at time $t$. With a one-timestep backward pass (to match the one-timestep value backups in the TD rule), this is:

$$
\beta_{s,t} = P(\mathbf{s}_t = s, \phi_t = 1 | \mathbf{o}_1 \ldots \mathbf{o}_{t+1})
$$

By Bayes' theorem, $\beta_{s,t} \propto P(\mathbf{o}_{t+1} | \mathbf{s}_t = s, \phi_t = 1) \cdot P(\mathbf{s}_t = s, \phi_t = 1 | \mathbf{o}_1 \ldots \mathbf{o}_t)$. The first term can be computed by integrating over $\mathbf{s}_{t+1}$ in the model: $P(\mathbf{o}_{t+1} | \mathbf{s}_t = s, \phi_t = 1) = \sum_{s' \in \mathcal{S}} Q_{ss'} \cdot O_{s'\mathbf{o}_{t+1}}$; the second requires integrating over possible state sequences and dwell times:

$$
P(\mathbf{s}_t = s, \phi_t = 1 | \mathbf{o}_1 \ldots \mathbf{o}_t) = \sum_{\tau=1}^{d_{lastO}} D_{s\tau} \cdot O_{s\mathbf{o}_{t-\tau+1}} \cdot P(\mathbf{s}_{t-\tau+1} = s, \phi_{t-\tau} = 1 | \mathbf{o}_1 \ldots \mathbf{o}_{t-\tau})
$$

where $d_{lastO}$ is the number of timesteps since the last non-null observation and $P(\mathbf{s}_{t-\tau+1} = s, \phi_{t-\tau} = 1 | \mathbf{o}_1 \ldots \mathbf{o}_{t-\tau})$, the chance that the process entered $s$ at $t - \tau + 1$, equals $\sum_{s' \in \mathcal{S}} Q_{s's} \cdot P(\mathbf{s}_{t-\tau} = s', \phi_{t-\tau} = 1 | \mathbf{o}_1 \ldots \mathbf{o}_{t-\tau})$, allowing recursive computation.

$\beta$ is used for TD learning because it represents the probability of a transition, which is the event that triggers a value update in fully observable semi-Markov TD. Due to partial observability, we may not be certain when transitions have occurred or from which states, so we perform TD updates to every state at every timestep, weighted by $\beta$. We denote our estimate of the value of state $s$ as $\hat{V}_s$, to distinguish it from the true value $V_s$. The update to $\hat{V}_s$ at time $t$ is proportional to the TD error:

$$\delta_{s,t} = \beta_{s,t}(E[\gamma^\tau] \cdot (\mathbf{r}_{t+1} + E[\hat{V}_{s'}]) - \hat{V}_s)$$

where $E[\gamma^\tau] = \sum_k \gamma^k P(\tau_t = k | \mathbf{s}_t = s, \phi_t = 1, \mathbf{o}_1 \ldots \mathbf{o}_{t+1})$ is the expected discounting (since dwell time may be uncertain) and $E[\hat{V}_{s'}] = \sum_{s' \in \mathcal{S}} \hat{V}_{s'} P(\mathbf{s}_{t+1} = s' | \mathbf{s}_t = s, \phi_t = 1, \mathbf{o}_{t+1})$ is the expected subsequent value. Both expectations are conditioned on the process having left state $s$ at time $t$, and computed using the internal world model.

As in previous models, we associate the error signal $\delta$ with DA activity. However, because of uncertainty as to the state of the world, the TD error signal is vector-valued rather than scalar. DA neurons could code this vector in a distributed manner, which might explain experimentally observed response variability between neurons [7]. Alternatively, $\delta_{s,t}$ can be approximated with a scalar, which performs well if the inferred state occupancy is sharply peaked. In our figures, we use such an approximation, plotting DA activity as the cumulative TD error over states (implicitly weighted by $\beta$): $\delta_t = \sum_{s \in \mathcal{S}} \delta_{s,t}$. An approximate version of the vector signal could be reconstructed at target areas by multiplying by $\beta_{s,t} / \sum_{s' \in \mathcal{S}} \beta_{s',t}$.

Note that with full observability, the (vector) learning rule reduces to standard semi-Markov TD, and conversely with full *un*observability, it nudges states in the direction of a value iteration backup. In fact, the algorithm is exact in that it has the same fixed point as value iteration, assuming the inference model matches the contingencies of the world. (Due to uncertainty it does so only approximately in our simulations.) We sketch the proof. With each TD update, $\hat{V}_s$ is nudged toward some target value with some step size $\beta_{s,t}$; the fixed point is the average of the targets, weighted by their probabilities and their step sizes. Fixing some arbitrary $t$, the update targets and $\beta$ are functions of the observations $\mathbf{o}_1 \ldots \mathbf{o}_{t+1}$, which are generated according to $P(\mathbf{o}_1 \ldots \mathbf{o}_{t+1})$. The fixed point is:

$$\hat{V}_s = \frac{\sum_{\mathbf{o}_1 \ldots \mathbf{o}_{t+1}} P(\mathbf{o}_1 \ldots \mathbf{o}_{t+1}) \cdot \beta_{s,t} \cdot E[\gamma^\tau] \cdot (\mathbf{r}_{t+1} + E[\hat{V}_{s'}])}{\sum_{\mathbf{o}_1 \ldots \mathbf{o}_{t+1}} P(\mathbf{o}_1 \ldots \mathbf{o}_t) \cdot \beta_{s,t}}$$

Marginalizing out the observations reduces this to Bellman's equation for $\hat{V}_s$, which is also, of course, the fixed-point equation for value iteration.

## 4 Results

When expected reward is delivered early, the semi-Markov model assumes that this signals an early transition into the ITI state, and it thus does not expect further reward or produce spurious negative error (Figure 1d, top). Because of variability in the model's ISI estimate, an early transition, while improbable, better explains the data than some other path through the state space. The early reward is worth more than expected, due to reduced discounting, and is thus accompanied by positive error.

The model can also infer a state transition from the passage of time, absent any observations. In Figure 1d (bottom), when the reward is delivered late, the system infers that the world has entered the ITI state without reward, producing negative error.

Figure 1f shows our model's behavior when the ISI is uniformly distributed [11]. (The dwell time distribution $D$ in the inference model was changed to reflect this distribution,

as an animal should learn a different model here.) Earlier-than-average rewards are worth more than expected (due to discounting) and cause positive prediction error, while later-than-average rewards cause negative error because they are more heavily discounted. This is broadly consistent with the experimental finding of decreasing response with increasing delay [11]. Inhibition at longer delays has not so far been observed in this experiment, though inhibition is in general difficult to detect. If discovered, such inhibition would support the semi-Markov model.

Because it combines a conditional probability model with TD learning, our approach can incorporate insights from previous behavioral theories into a physiological model. Our state inference approach is based on a hidden Markov model (HMM) account we previously advanced to explain animal learning about the temporal relationships of events [15]. The present theory (with the model learning scheme from that paper) would account for the same data. Our model also accommodates two important theoretical ideas from more abstract models of animal learning that previous TD models cannot. One is the notion of *uncertainty* in some of its internal parameters, which Kakade and Dayan [16] use to explain interval timing and attentional effects in learning. Second, Gallistel has suggested that animal learning processes are *timescale invariant*. For example, altering the speed of events has no effect on the number of trials it takes animals to learn a stimulus-reward association [18]. This is not true of Markov TD models because their transitions are clocked to a fixed timescale. With tapped delay lines, timescale dilation increases the number of marker states in Figure 1a and slows learning. But our semi-Markov model is timescale invariant: learning is induced by state transitions which in turn are triggered by events or by the passage of time on a scale controlled by the internal model. (The form of temporal discounting we use is not timescale invariant, but this can be corrected as in [5].)

## 5   Discussion

We have presented a model of the DA system that improves on previous models' accounts of data involving temporal variability and partial observability, because, unlike prior models, it is grounded in a formalism that explicitly incorporates these considerations. Like previous models, ours identifies the DA response with reward prediction error, but it differs in the representational systems driving the predictions. Previous models assumed that tapped delay lines transcribed raw sensory events; ours envisions that these events inform a more active process of inference about the underlying state of the world. This is a principled approach to the problem of representing state when events can be separated by delays.

Simpler schemes may capture the neuronal data, which are sparse, but without addressing the underlying computational issues we identify, they are unlikely to generalize. For instance, Suri and Schultz [4] propose that reward delivery overrides stimulus representations, canceling pending predictions and eliminating the spurious negative error in Figure 1c (top). But this would disrupt the behaviorally demonstrated ability of animals to learn that a stimulus predicts a *series* of rewards. Such static representational rules are insufficient since different tasks have different mnemonic requirements. In our account, unlike more ad-hoc theories, the problem of learning an appropriate representation for a task is well specified: it is the problem of modeling the task. Though we have not simulated model learning here (this is an important area for future work), it is possible using online HMM learning, and we have used this technique in a model of conditioning [15]. Another issue for the future is extending our theory to encompass action selection. DA models often assume an actor-critic framework [1] in which reward predictions are used to evaluate action selection policies. Partial observability complicates such an extension here, since policies must be defined over belief states (*distributions* over the hidden states $\mathcal{S}$) to accommodate uncertainty; our use of $\mathcal{S}$ as a linear basis for value predictions is thus an oversimplification.

Puzzlingly, the data we consider suggest that animals build internal models but also use

sample-based TD methods to predict values. Given a full world model (which could in principle be solved directly for $V$), it seems unclear why TD learning should be necessary. But since the world model must be learned incrementally online, it may be infeasible to continually re-solve it, and parts of the model may be poorly specified. In this case, TD learning in the inferred state space could maintain a reasonably current and observationally grounded value function. (Our particular formulation, which relies extensively on the model in the TD rule, may not be ideal from this perspective.)

Suri [19] and Dayan [20] have also proposed TD theories of DA that incorporate world models to explain behavioral effects, though they do not address the theoretical issues or dopaminergic data considered here. While those accounts use the world model for directly anticipating future events, we have proposed another role for it in state inference. Also unlike our theory, the others cannot explain the experiments discussed in [15] because their internal models cannot represent simultaneous or backward contingencies. However, they treat the two major issues we have neglected: world model learning and action planning.

The formal models in question have roughly equivalent explanatory power: a semi-Markov model can be simulated (to arbitrarily fine temporal discretization) by a Markov model that subdivides its states by dwell time. There is also an isomorphism between higher-order and partially observable Markov models. Thus it would be possible to devise a state representation for a Markov model that copes properly with temporal variability. But doing so by elaborating the tapped delay line architecture would amount to building a clockwork engine for the inference process we describe, without the benefit of useful abstractions such as distributions over intervals; a clearer approach would subdivide the states in our model.

Though there exist isomorphisms between the formal models, there are algorithmic differences that may make our proposal experimentally distinguishable from others. The inhibitory responses in Figure 1f reflect the way semi-Markov models account for the costs of delays; they would not be seen in a Markov model with subdivided states. Such inhibition is somewhat parameter-dependent, since if inference parameters assign high probability to unsignaled transitions the decrease in reward value with delay can be mitigated by increasing uncertainty about the hidden state. Nonetheless, should data not uphold our prediction of inhibitory responses to late rewards, they would suggest a different definition of a state's value. One choice would be the subdivision of our semi-Markov states by dwell time discussed above, which in the experiment of Figure 1f would decrease TD error toward but not past zero for longer delays. In this case, later rewards are less surprising because the conditional probability of reward increases as time passes without reward.

A related prediction suggested by our model is that DA responses not just to rewards but also to stimuli that signal reward might be modulated by their timing relative to expectation. Responses to reward-predicting stimuli disappear in overtrained animals, presumably because the stimuli come to be predicted by events in the previous trial [7]. In tapped delay line models, this is possible only for a constant ITI (since if expectancy is divided between a number of states, stimulus delivery in any one of them cannot be completely predicted away). But the response to a stimulus in the semi-Markov model can show behavior exactly analogous to the reward response in Figure 1f — positive or negative error depending on the time of delivery relative to expectation. So, even in an experiment involving a randomized ITI, the net stimulus response (averaged over the range of ITIs) could be attenuated. Such behavior occurred in our simulations; the modeled DA responses to the stimuli in Figures 1d and 1f are positive because they were taken after shorter-than-average ITIs. It is difficult to evaluate this observation against available data, since the experiment involving overtrained monkeys [7] contained minimal ITI variability.

We have suggested that the TD error may be a vector signal, with different neurons signaling errors for different elements of a state distribution. This could be investigated experimentally by recording DA neurons as a situation of ambiguous reward expectancy (e.g. one

reward or three) resolved into a situation of intermediate, determinate reward expectancy (e.g. two rewards). Neurons carrying an aggregate error should uniformly report no error, but with a vector signal, different neurons might report both positive and negative error.

**Acknowledgments**

This work was supported by National Science Foundation grants IIS-9978403 and DGE-9987588. Aaron Courville was funded in part by a Canadian NSERC PGS B fellowship. We thank Sham Kakade and Peter Dayan for helpful discussions.

## References

[1] JC Houk, JL Adams, and AG Barto. A model of how the basal ganglia generate and use neural signals that predict reinforcement. In JC Houk, JL Davis, and DG Beiser, editors, *Models of Information Processing in the Basal Ganglia*, pages 249–270. MIT Press, 1995.

[2] PR Montague, P Dayan, and TJ Sejnowski. A framework for mesencephalic dopamine systems based on predictive Hebbian learning. *J Neurosci*, 16:1936–1947, 1996.

[3] W Schultz, P Dayan, and PR Montague. A neural substrate of prediction and reward. *Science*, 275:1593–1599, 1997.

[4] RE Suri and W Schultz. A neural network with dopamine-like reinforcement signal that learns a spatial delayed response task. *Neurosci*, 91:871–890, 1999.

[5] ND Daw and DS Touretzky. Long-term reward prediction in TD models of the dopamine system. *Neural Comp*, 14:2567–2583, 2002.

[6] RS Sutton. Learning to predict by the method of temporal differences. *Machine Learning*, 3:9–44, 1988.

[7] W Schultz. Predictive reward signal of dopamine neurons. *J Neurophys*, 80:1–27, 1998.

[8] RS Sutton and AG Barto. Time-derivative models of Pavlovian reinforcement. In M Gabriel and J Moore, editors, *Learning and Computational Neuroscience: Foundations of Adaptive Networks*, pages 497–537. MIT Press, 1990.

[9] JR Hollerman and W Schultz. Dopamine neurons report an error in the temporal prediction of reward during learning. *Nature Neurosci*, 1:304–309, 1998.

[10] DS Touretzky, ND Daw, and EJ Tira-Thompson. Combining configural and TD learning on a robot. In *ICDL 2*, pages 47–52. IEEE Computer Society, 2002.

[11] CD Fiorillo and W Schultz. The reward responses of dopamine neurons persist when prediction of reward is probabilistic with respect to time or occurrence. In *Soc. Neurosci. Abstracts*, volume 27: 827.5, 2001.

[12] LP Kaelbling, ML Littman, and AR Cassandra. Planning and acting in partially observable stochastic domains. *Artif Intell*, 101:99–134, 1998.

[13] SJ Bradtke and MO Duff. Reinforcement learning methods for continuous-time Markov Decision Problems. In *NIPS 7*, pages 393–400. MIT Press, 1995.

[14] L Chrisman. Reinforcement learning with perceptual aliasing: The perceptual distinctions approach. In *AAAI 10*, pages 183–188, 1992.

[15] AC Courville and DS Touretzky. Modeling temporal structure in classical conditioning. In *NIPS 14*, pages 3–10. MIT Press, 2001.

[16] S Kakade and P Dayan. Acquisition in autoshaping. In *NIPS 12*, pages 24–30. MIT Press, 2000.

[17] Y Guedon and C Cocozza-Thivent. Explicit state occupancy modeling by hidden semi-Markov models: Application of Derin's scheme. *Comp Speech and Lang*, 4:167–192, 1990.

[18] CR Gallistel and J Gibbon. Time, rate and conditioning. *Psych Rev*, 107(2):289–344, 2000.

[19] RE Suri. Anticipatory responses of dopamine neurons and cortical neurons reproduced by internal model. *Exp Brain Research*, 140:234–240, 2001.

[20] P Dayan. Motivated reinforcement learning. In *NIPS 14*, pages 11–18. MIT Press, 2001.
